# Harmony Networks Do Not Work

**René Gourley**
School of Computing Science
Simon Fraser University
Burnaby, B.C., V5A 1S6, Canada
gourley@mprgate.mpr.ca

## Abstract

Harmony networks have been proposed as a means by which connectionist models can perform symbolic computation. Indeed, proponents claim that a harmony network can be built that constructs parse trees for strings in a context free language. This paper shows that harmony networks do not work in the following sense: they construct many outputs that are not valid parse trees.

In order to show that the notion of systematicity is compatible with connectionism, Paul Smolensky, Geraldine Legendre and Yoshiro Miyata (Smolensky, Legendre, and Miyata 1992; Smolensky 1993; Smolensky, Legendre, and Miyata 1994) proposed a mechanism, "Harmony Theory," by which connectionist models purportedly perform structure sensitive operations without implementing classical algorithms. Harmony theory describes a "harmony network" which, in the course of reaching a stable equilibrium, apparently computes parse trees that are valid according to the rules of a particular context-free grammar.

Harmony networks consist of four major components which will be explained in detail in Section 1. The four components are,

**Tensor Representation:** A means to interpret the activation vector of a connectionist system as a parse tree for a string in a context-free language.

**Harmony:** A function that maps all possible parse trees to the non-positive integers so that a parse tree is valid if and only if its harmony is zero.

**Energy:** A function that maps the set of activation vectors to the real numbers and which is minimized by certain connectionist networks[1].

**Recursive Construction:** A system for determining the weight matrix of a connectionist network so that if its activation vector is interpreted as a parse

tree, then the network's energy is the negation of the harmony of that parse tree.

Smolensky et al. contend that, in the process of minimizing their energy values, harmony networks implicitly maximize the harmony of the parse tree represented by their activation vector. Thus, if the harmony network reaches a stable equilibrium where the energy is equal to zero, the parse tree that is represented by the activation vector must be a valid parse tree:

> When the lower-level description of the activation-spreading process satisfies certain mathematical properties, this process can be analyzed on a higher level as the construction of that structure including the given input structure which *maximizes Harmony*. (Smolensky 1993, p848, emphasis is original)

Unfortunately, harmony networks do not work — they do not always construct maximum-harmony parse trees. The problem is that the energy function is defined on the values of the activation vector. By contrast, the harmony function is defined on possible parse trees. Section 2 of this paper shows that these two domains are not equal, that is, there are some activation vectors that do not represent any parse tree.

The recursive construction merely guarantees that the energy function passes through zero at the appropriate points; its minima are unrestricted. So, while it may be the case that the energy and harmony functions are negations of one another, it is not always the case that a local minimum of one is a local maximum of the other. More succinctly, the harmony network will find minima that are not even trees, let alone valid parse trees.

The reason why harmony networks do not work is straightforward. Section 3 shows that the weight matrix must have only negative eigenvalues, for otherwise the network constructs structures which are not valid trees. Section 4 shows that if the weight matrix has only negative eigenvalues, then the energy function admits only a single zero — the origin. Furthermore, we show that the origin cannot be interpreted as a valid parse tree. Thus, the stable points of a harmony network are not valid parse trees.

# 1  HARMONY NETWORKS

## 1.1  TENSOR REPRESENTATION

Harmony theory makes use of tensor products (Smolensky 1990; Smolensky, Legendre, and Miyata 1992; Legendre, Miyata, and Smolensky 1991) to convolve symbols with their roles. The resulting products are then added to represent a labelled tree using the harmony network's activation vector. The particular tensor product used is very simple:

$$(a_1, a_2, \ldots, a_n) \otimes (b_1, b_2, \ldots, b_m) =$$
$$(a_1 b_1, a_1 b_2, \ldots, a_1 b_m, a_2 b_1, a_2 b_2, \ldots, a_2 b_m, \ldots, a_n b_m)$$

If two tensors of differing dimensions are to be added, then they are essentially concatenated.

Binary trees are represented with this tensor product using the following recursive rules:

1. The tensor representation of a tree containing no vertices is 0.

Table 1: Rules for determining harmony and the weight matrix. Let $G = (V, \Sigma, P, S)$ be a context-free grammar of the type suggested in section 1.2. The rules for determining the harmony of a tree labelled with $V$ and $\Sigma$ are shown in the second column. The rules for determining the system of equations for recursive construction are shown in the third column. (Smolensky, Legendre, and Miyata 1992; Smolensky 1993)

| Grammar Element | Harmony Rule | Energy Equation |
|---|---|---|
| $S$ | For every node labelled $S$ add -1 to $H(T)$. | Include $(S + \vec{0} \otimes r_l) W_{root} (S + \vec{0} \otimes r_l) = 2$ in the system of equations |
| $x \in \Sigma$ | For every node labelled $x$ add -1 to $H(T)$. | Include $(x + \vec{0} \otimes r_l) W_{root} (x + \vec{0} \otimes r_l) = 2$ in the system of equations |
| $x \in V \setminus \{S\}$ | For every node labelled $x$ add -2 or -3 to $H(T)$ depending on whether or not $x$ appears on the left of a production with two symbols on the right. | Include $(x + \vec{0} \otimes r_l) W_{root} (x + \vec{0} \otimes r_l) = 4$ or 6 in the system of equations, depending on whether or not $x$ appears on the left of a production with two symbols on the right. |
| $x \rightarrow yz$ or $x \rightarrow y \in P$ | For every edge where $x$ is the parent and $y$ is the left child add 2. Similarly, add 2 every time $z$ is the right child of $x$. | Include in the system of equations, $(x + \vec{0} \otimes r_l) W_{root} (\vec{0} + y \otimes r_l) = -2$ $(\vec{0} + y \otimes r_l) W_{root} (x + \vec{0} \otimes r_l) = -2$ $(x + \vec{0} \otimes r_l) W_{root} (\vec{0} + z \otimes r_l) = -2$ $(\vec{0} + z \otimes r_l) W_{root} (x + \vec{0} \otimes r_l) = -2$ |

2. If $A$ is the root of a tree, and $T_L, T_R$ are the tensor product representations of its left subtree and right subtree respectively, then $A + T_L \otimes r_l + T_R \otimes r_r$ is the tensor representation of the whole tree.

The vectors, $r_l$, and $r_r$ are called "role vectors" and indicate the roles of left child and right child.

## 1.2  HARMONY

Harmony (Legendre, Miyata, and Smolensky 1990; Smolensky, Legendre, and Miyata 1992) describes a way to determine the well-formedness of a potential parse tree with respect to a particular context free grammar. Without loss of generality, we can assume that the right-hand side of each production has at most two symbols, and if a production has two symbols on the right, then it is the only production for the variable on its left side. For a given binary tree, $T$, we compute the harmony of $T$, $H(T)$ by first adding the negative contributions of all the nodes according to their labels, and then adding the contributions of the edges (see first two columns of table 1).

## 1.3    ENERGY

Under certain conditions, some connectionist models are known to admit the following energy or Lyapunov function (see Legendre, Miyata, and Smolensky 1991):

$$E(a) = -\frac{1}{2}a^t W a$$

Here, $W$ is the weight matrix of the connectionist network, and $a$ is its activation vector. Every non-equilibrium change in the activation vector results in a strict decrease in the network's energy. In effect, the connectionist network serves to minimize its energy as it moves towards equilibrium.

## 1.4    RECURSIVE CONSTRUCTION

Smolensky, Legendre, and Miyata (1992) proposed that the recursive structure of their tensor representations together with the local nature of the harmony calculation could be used to construct the weight matrix for a network whose energy function is the negation of the harmony of the tree represented by the activation vector. First construct a matrix $W_{root}$ which satisfies a system of equations. The system of equations is found by including equations for every symbol and production in the grammar, as shown in column three of table 1. Gourley (1995) shows that if $W$ is constructed from copies of $W_{root}$ according to a particular formula, and if $a_T$ is a tensor representation for a tree, $T$, then $E(a_T) = -H(T)$.

## 2    SOME ACTIVATIONS ARE NOT TREES

As noted above, the reason why harmony networks do not work is that they seek minima in their state space which may not coincide with parse tree representations. One way to amelioarate this would be to make every possible activation vector represent some parse tree. If every activation vector represents some parse tree, then the rules that determine the weight matrix will ensure that the energy minima agree with the valid parse trees. Unfortunately, in that case, the system of equations used to determine $W_{root}$ has no solution.

If every activation vector is to represent some parse tree, and the symbols of the grammar are two dimensional, then there are symbols represented by each vector, $(x_1, x_1), (x_1, x_2), (x_2, x_1)$, and $(x_2, x_2)$, where $x_1 \neq x_2$. These symbols must satisfy the equations given in table 1 , and so,

$$\left. \begin{array}{rcl} x_1^2(W_{root_{11}} + W_{root_{12}} + W_{root_{21}} + W_{root_{22}}) & = & h_1 \\ x_1^2 W_{root_{11}} + x_1 x_2 W_{root_{12}} + x_1 x_2 W_{root_{21}} + x_2^2 W_{root_{22}} & = & h_2 \\ x_2^2 W_{root_{11}} + x_1 x_2 W_{root_{12}} + x_1 x_2 W_{root_{21}} + x_1^2 W_{root_{22}} & = & h_3 \\ x_2^2(W_{root_{11}} + W_{root_{12}} + W_{root_{21}} + W_{root_{22}}) & = & h_4 \end{array} \right\} \begin{array}{l} \text{where } h_i \in \\ \{2, 4, 6\} \end{array}$$

Because $h_i \in \{2, 4, 6\}$, there must be a pair $h_i, h_j$ which are equal. In that case, it can be shown using Gaussian elimination that there is no solution for $W_{root_{11}}, W_{root_{12}}, W_{root_{21}}, W_{root_{22}}$. Similarly, if the symbols are represented by vectors of dimension three or greater, the same contradiction occurs.

Thus there are some activation vectors that do not represent any tree — valid or invalid. The question now becomes one of determining whether all of the harmony network's stable equilibria are valid parse trees.

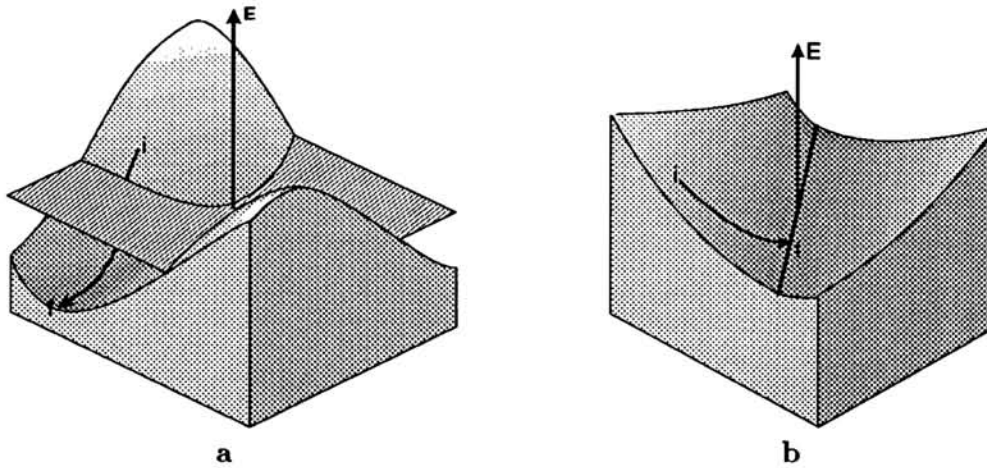

Figure 1: Energy functions of two-dimensional harmony networks. In each case, the points $i$ and $f$ respectively represent an initial and a final state of the network. In **a**, one eigenvector is positive and the other is negative; the hashed plane represents the plane $E = 0$ which intersects the energy function and the vertical axis at the origin. In **b**, one eigenvalue is negative while the other is zero; The heavy line represents the intersection of the surface with the plane $E = 0$ and it intersects the vertical axis at the origin.

## 3   NON-NEGATIVE EIGENVECTORS YIELD NON-TREES

If any of the eigenvalues of the weight matrix, $W$, is positive, then it is easy to show that the harmony network will seek a stable equilibrium that does not represent a parse tree at all. Let $\lambda > 0$ be a positive eigenvalue of $W$, and let $e$ be an eigenvector, corresponding to $\lambda$, that falls within the state space. Then,

$$E(e) = -\frac{1}{2}e^t We = -\frac{1}{2}\lambda e^t e < 0.$$

Because the energy drops below zero, the harmony network would have to undergo an energy increase in order to find a zero-energy stable equilibrium. This cannot happen, and so, the network reaches an equilibrium with energy strictly less than zero.

Figure 1a illustrates the energy function of a harmony network where one eigenvalue is positive. Because harmony is the negation of energy, in this figure all the valid parse trees rest on the hashed plane, and all the invalid parse trees are above it. As we can see, the harmony network with positive eigenvalues will certainly find stable equilibria which are not valid parse tree representations.

Now, suppose $W$, the weight matrix, has a zero eigenvalue. If $e$ is an eigenvector corresponding to that eigenvalue, then for every real $\alpha$, $\alpha We = 0$. Consequently, one of the following must be true:

1. $\alpha e$ is not a stable equilibrium. In that case, the energy function must drop below zero, yielding a sub-zero stable equilibrium — a stable equilibrium that does not represent any tree.

2. $\alpha e$ is a stable equilibrium. Then for every $\alpha$, $\alpha e$ must be a valid tree representation. Such a situation is represented in fig-

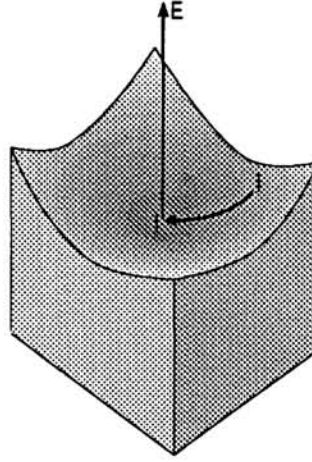

Figure 2: The energy function of a two-dimensional harmony network where both eigenvalues are negative. The vertical axis pierces the surface at the origin, and the points $i$ and $f$ respectively represent an initial and a final state of the network.

ure 1b where the set of all points $\alpha e$ is represented by the heavy line. This implies that there is a symbol, $(a_1, a_2, \ldots, a_n)$, such that $\alpha_1(a_1, a_2, \ldots, a_n), \alpha_2(a_1, a_2, \ldots, a_n), \ldots, \alpha_{n^2+1}(a_1, a_2, \ldots, a_n)$ are also all symbols. As before, this implies that $W_{root}$ must satisfy the equation,

$$((a_1, \ldots, a_n) + \vec{0} \otimes r_l)^t W_{root}((a_1, \ldots, a_n) + \vec{0} \otimes r_l) = \frac{h_i}{\alpha_i^2}, \; h_i \in \{2, 4, 6\}$$

for $i = 1 \ldots n^2 + 1$. Again using Gaussian elimination, it can be shown that there is no solution to this system of equations.

In either case, the harmony network admits stable equilibria that do not represent any tree. Thus, the eigenvalues must all be negative.

## 4  NEGATIVE EIGENVECTORS YIELD NON-TREES

If all the eigenvalues of the weight matrix are negative, then the energy function has a very special shape: it is a paraboloid centered on the origin and concave in the direction of positive energy. This is easily seen by considering the first and second derivatives of $E$:

$$\frac{\partial E(\vec{x})}{\partial x_i} = -\sum_j W_{i,j} x_i \qquad \qquad \frac{\partial^2 E(\vec{x})}{\partial x_i \partial x_j} = -W_{i,j}$$

Clearly, all the first derivatives are zero at the origin, and so, it is a critical point. Now the origin is a strict minimum if all the roots of the following well-known equation are positive:

$$0 = \det \begin{vmatrix} \frac{\partial^2 E(\vec{x})}{\partial x_1 \partial x_1} - \lambda & \frac{\partial^2 E(\vec{x})}{\partial x_1 \partial x_2} & \cdots \\ \frac{\partial^2 E(\vec{x})}{\partial x_2 \partial x_1} & \frac{\partial^2 E(\vec{x})}{\partial x_1 \partial x_2} - \lambda & \\ \vdots & & \ddots \end{vmatrix} = \det |-W - \lambda I|$$

$\det |-W - \lambda I|$ is the characteristic polynomial of $-W$. If $\lambda$ is a root then it is an eigenvalue of $-W$, or equivalently, it is the negative of an eigenvalue of $W$. Because all of $W$'s eigenvalues are negative, the origin is a strict minimum, and indeed it is the only minimum. Such a harmony network is illustrated in Figure 2.

Thus the origin is the only stable point where the energy is zero, but it cannot represent a parse tree which is valid for the grammar. If it does, then

$$S + T_L \otimes r_l + T_R \otimes r_r = (0, \ldots, 0)$$

where $T_L, T_R$ are appropriate left and right subtree representations, and $S$ is the start symbol of the grammar. Because each of the subtrees is multiplied by either $r_l$ or $r_r$, they are not the same dimension as $S$, and are consequently concatenated instead of added. Therefore $S = \vec{0}$. But then, $W_{root}$ must satisfy the equation

$$(\vec{0} + \vec{0} \otimes r_l)W_{root}(\vec{0} + \vec{0} \otimes r_l) = -2$$

This is impossible, and so, the origin is not a valid tree representation.

## 5  CONCLUSION

This paper has shown that in every case, a harmony network will reach stable equilibria that are not valid parse trees. This is not unexpected. Because the energy function is a very simple function, it would be more surprising if such a connectionist system could construct complicated structures such as parse trees for a context free grammar.

### Acknowledgements

The author thanks Dr. Robert Hadley and Dr. Arvind Gupta, both of Simon Fraser University, for their invaluable comments on a draft of this paper.

## Footnotes

[1]Smolensky, Legendre and Miyata use the term "harmony" to refer to both energy and harmony. To distinguish between them, we will use the term that is often used to describe the Lyapunov function of dynamic systems, "energy" (see for example Golden 1986).

## References

Golden, R. (1986). The 'brain-state-in-a-box' neural model is a gradient descent algorithm. *Journal of Mathematical Psychology 30*, 73–80.

Gourley, R. (1995). Tensor represenations and harmony theory: A critical analysis. Master's thesis, Simon Fraser University, Burnaby, Canada. In preparation.

Legendre, G., Y. Miyata, and P. Smolensky (1990). Harmonic grammar – a formal multi-level connectionist theory of linguistic well-formedness: Theoretical foundations. In *Proceedings of the Twelfth National Conference on Cognitive Science*, Cambridge, MA, pp. 385–395. Lawrence Erlbaum.

Legendre, G., Y. Miyata, and P. Smolensky (1991). Distributedrecursive structure processing. In B. Mayoh (Ed.), *Proceedings of the 1991 Scandinavian Conference on Artificial Intelligence*, Amsterdam, pp. 47–53. IOS Press.

Smolensky, P. (1990). Tensor product variable binding and the representation of symbolic structures in connectionist systems. *Artificial Intelligence 46*, 159–216.

Smolensky, P. (1993). Harmonic grammars for formal languages. In S. Hanson, J. Cowan, and C. Giles (Eds.), *Advances in Neural Information Processing Systems 5*, pp. 847–854. San Mateo: Morgan Kauffman.

Smolensky, P., G. Legendre, and Y. Miyata (1992). Principles for an integrated connectionist/symbolic theory of higher cognition. Technical Report CU-CS-600-92, University of Colorado Computer Science Department.

Smolensky, P., G. Legendre, and Y. Miyata (1994). Integrating connectionist and symbolic computation for the theory of language. In V. Honavar and L. Uhr (Eds.), *Artificial Intelligence and Neural Networks: Steps Toward Principled Integration*, pp. 509–530. Boston: Academic Press.
